# Identity Uncertainty and Citation Matching

**Hanna Pasula, Bhaskara Marthi, Brian Milch, Stuart Russell, Ilya Shpitser**
Computer Science Division, University Of California
387 Soda Hall, Berkeley, CA 94720-1776
`pasula, marthi, milch, russell, ilyas@cs.berkeley.edu`

## Abstract

*Identity uncertainty* is a pervasive problem in real-world data analysis. It arises whenever objects are not labeled with unique identifiers or when those identifiers may not be perceived perfectly. In such cases, two observations may or may not correspond to the same object. In this paper, we consider the problem in the context of *citation matching*—the problem of deciding which citations correspond to the same publication. Our approach is based on the use of a relational probability model to define a generative model for the domain, including models of author and title corruption and a probabilistic citation grammar. Identity uncertainty is handled by extending standard models to incorporate probabilities over the possible mappings between terms in the language and objects in the domain. Inference is based on Markov chain Monte Carlo, augmented with specific methods for generating efficient proposals when the domain contains many objects. Results on several citation data sets show that the method outperforms current algorithms for citation matching. The declarative, relational nature of the model also means that our algorithm can determine object characteristics such as author names by combining multiple citations of multiple papers.

## 1   INTRODUCTION

Citation matching is the problem currently handled by systems such as Citeseer [1].[1] Such systems process a large number of scientific publications to extract their citation lists. By grouping together all co-referring citations (and, if possible, linking to the actual cited paper), the system constructs a database of "paper" entities linked by the "$cites(p_1, p_2)$" relation. This is an example of the general problem of determining the existence of a set of objects, and their properties and relations, given a collection of "raw" perceptual data; this problem is faced by intelligence analysts and intelligent agents as well as by citation systems.

A key aspect of this problem is determining when two observations describe the same object; only then can evidence be combined to develop a more complete description of the object. Objects seldom carry unique identifiers around with them, so *identity uncertainty* is ubiquitous. For example, Figure 1 shows two citations that probably refer to the same paper, despite many superficial differences. Citations appear in many formats and are rife with errors of all kinds. As a result, Citeseer—which is specifically designed to overcome such problems—currently lists more than 100 distinct AI textbooks published by Russell

[Lashkari et al 94] Collaborative Interface Agents, Yezdi Lashkari, Max Metral, and Pattie Maes, Proceedings of the Twelfth National Conference on Articial Intelligence, MIT Press, Cambridge, MA, 1994.

Metral M. Lashkari, Y. and P. Maes. Collaborative interface agents. In Conference of the American Association for Artificial Intelligence, Seattle, WA, August 1994.

Figure 1: Two citations that probably refer to the same paper.

and Norvig on or around 1995, from roughly 1000 citations. Identity uncertainty has been studied independently in several fields. *Record linkage* [2] is a method for matching up the records in two files, as might be required when merging two databases. For each pair of records, a *comparison vector* is computed that encodes the ways in which the records do and do not match up. EM is used to learn a naive-Bayes distribution over this vector for both matched and unmatched record pairs, so that the pairwise match probability can then be calculated using Bayes' rule. Linkage decisions are typically made in a greedy fashion based on closest match and/or a probability threshold, so the overall process is order-dependent and may be inconsistent. The model does not provide for a principled way to combine matched records. A richer probability model is developed by Cohen *et al* [3], who model the database as a combination of some "original" records that are correct and some number of erroneous versions. They give an efficient greedy algorithm for finding a single locally optimal assignment of records into groups.

*Data association* [4] is the problem of assigning new observations to existing trajectories when multiple objects are being tracked; it also arises in robot mapping when deciding if an observed landmark is the same as one previously mapped. While early data association systems used greedy methods similar to record linkage, recent systems have tried to find high-probability global solutions [5] or to approximate the true posterior over assignments [6]. The latter method has also been applied to the problem of *stereo correspondence*, in which a computer vision system must determine how to match up features observed in two or more cameras [7]. Data association systems usually have simple observation models (e.g., Gaussian noise) and assume that observations at each time step are all distinct. More general patterns of identity occur in natural language text, where the problem of *anaphora resolution* involves determining whether phrases (especially pronouns) co-refer; some recent work [8] has used an early form of relational probability model, although with a somewhat counterintuitive semantics.

Citeseer is the best-known example of work on citation matching [1]. The system groups citations using a form of greedy agglomerative clustering based on a text similarity metric (see Section 6). McCallum *et al* [9] use a similar technique, but also develop clustering algorithms designed to work well with large numbers of small clusters (see Section 5).

With the exception of [8], all of the preceding systems have used domain-specific algorithms and data structures; the probabilistic approaches are based on a fixed probability model. In previous work [10], we have suggested a declarative approach to identity uncertainty using a formal language—an extension of relational probability models [11]. Here, we describe the first substantial application of the approach. Section 2 explains how to specify a generative probability model of the domain. The key technical point (Section 3) is that the possible worlds include not only objects and relations but also mappings from terms in the language to objects in the domain, and the probability model must include a prior over such mappings. Once the extended model has been defined, Section 4 details the probability distributions used. A general-purpose inference method is applied to the model. We have found Markov chain Monte Carlo (MCMC) to be effective for this and other applications (see Section 5); here, we include a method for generating effective proposals based on ideas from [9]. The system also incorporates an EM algorithm for learning the local probability models, such as the model of how author names are abbreviated, reordered, and misspelt in citations. Section 6 evaluates the performance of four datasets originally used to test the Citeseer algorithms [1]. As well as providing significantly better performance,

our system is able to reason simultaneously about papers, authors, titles, and publication types, and does a good job of extracting this information from the grouped citations. For example, an author's name can be identified more accurately by combining information from multiple citations of several different papers. The errors made by our system point to some interesting unmodeled aspects of the citation process.

## 2 RPMs

Reasoning about identity requires reasoning about objects, which requires at least some of the expressive power of a first-order logical language. Our approach builds on relational probability models (RPMs) [11], which let us specify probability models over possible worlds defined by objects, properties, classes, and relations.

### 2.1 Basic RPMs

At its most basic, an RPM, as defined by Koller *et al* [12], consists of

- A set $\mathcal{C}$ of *classes* denoting sets of objects, related by subclass/superclass relations.
- A set $\mathcal{I}$ of *named instances* denoting objects, each an instance of one class.
- A set $\mathcal{A}$ of *complex attributes* denoting functional relations. Each complex attribute $A$ has a domain type $Dom[A] \in \mathcal{C}$ and a range type $Range[A] \in \mathcal{C}$.
- A set $\mathcal{B}$ of *simple attributes* denoting functions. Each simple attribute $B$ has a domain type $Dom[B] \in \mathcal{C}$ and a range $Val[B]$.
- A set of conditional probability models $P(B|Pa[B])$ for the simple attributes. $Pa[B]$ is the set of $B$'s *parents*, each of which is a nonempty chain of (appropriately typed) attributes $\sigma = A_1. \cdots . A_n . B'$, where $B'$ is a simple attribute. Probability models may be attached to instances or inherited from classes. The parent links should be such that no cyclic dependencies are formed.
- A set of *instance statements*, which set the value of a complex attribute to an instance of the appropriate class.

We also use a slight variant of an additional concept from [11]: *number uncertainty*, which allows for multi-valued complex attributes of uncertain cardinality. We define each such attribute $A$ as a relation rather than a function, and we associate with it a simple attribute $\#[A]$ (i.e., the number of values of $A$) with a domain type $Dom[A]$ and a range $\{0, 1, \ldots, \max \#[A]\}$.

### 2.2 RPMs for citations

Figure 2 outlines an RPM for the example citations of Figure 1. There are four classes, the self-explanatory *Author*, *Paper*, and *Citation*, as well as *AuthorAsCited*, which represents not actual authors, but author names as they appear when cited. Each citation we wish to match leads to the creation of a *Citation* instance; instances of the remaining three classes are then added as needed to fill all the complex attributes. E.g., for the first citation of Figure 1, we would create a *Citation* instance $C_1$, set its *text* attribute to the string "Me-tral M. ...August 1994.", and set its *paper* attribute to a newly created *Paper* instance, which we will call $P_1$. We would then introduce $\max(\#[author])$ (here only 3, for simplicity) *AuthorAsCited* instances ($D_{11}$, $D_{12}$, and $D_{13}$) to fill the $P_1$.*obsAuthors* (i.e., observed authors) attribute, and an equal number of *Author* instances ($A_{11}$, $A_{12}$, and $A_{13}$) to fill both the $P_1$.*authors[i]* and the $D_{1i}$.*author* attributes. (The complex attributes would be set using instance statements, which would then also constrain the cited authors to be equal to the authors of the actual paper. [2]) Assuming (for now) that the value of $C_1$.*parse*

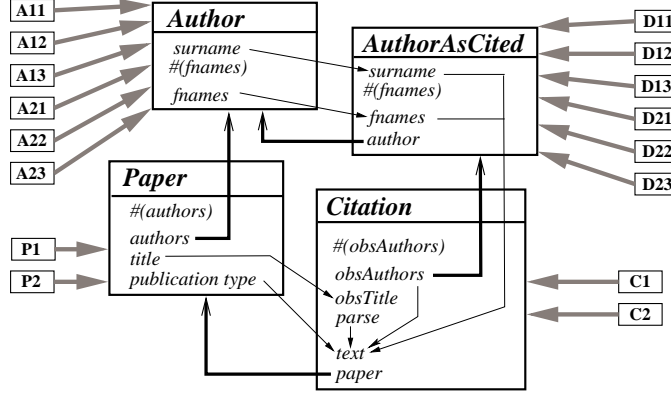

Figure 2: An RPM for our Citeseer example. The large rectangles represent classes: the dark arrows indicate the ranges of their complex attributes, and the light arrows lay out all the probabilistic dependencies of their basic attributes. The small rectangles represent instances, linked to their classes with thick grey arrows. We omit the instance statements which set many of the complex attributes.

is observed, we can set the values of all the basic attributes of the *Citation* and *AuthorAsCited* instances. (E.g., given the correct parse, $D_{11}$.*surname* would be set to Lashkari, and $D_{12}$.*fnames* would be set to (Max)). The remaining basic attributes — those of the *Paper* and *Author* instances — represent the "true" attributes of those objects, and their values are unobserved.

The standard semantics of RPMs includes the unique names assumption, which precludes identity uncertainty. Under this assumption, any two papers are assumed to be different unless we know for a fact that they are the same. In other words, although there are many ways in which the terms of the language can map to the objects in a possible world, only one of these identity mappings is legal: the one with the fewest co-referring terms. It is then possible to express the RPM as an equivalent Bayesian network: each of the basic attributes of each of the objects becomes a node, with the appropriate parents and probability model. RPM inference usually involves the construction of such a network. The Bayesian network equivalent to our RPM is shown in Figure 3.

## 3 IDENTITY UNCERTAINTY

In our application, any two citations may or may not refer to the same paper. Thus, for citations $C_1$ and $C_2$, there is uncertainty as to whether the corresponding papers $P_1$ and $P_2$ are in fact the same object. If they are the same, they will share one set of basic attributes;

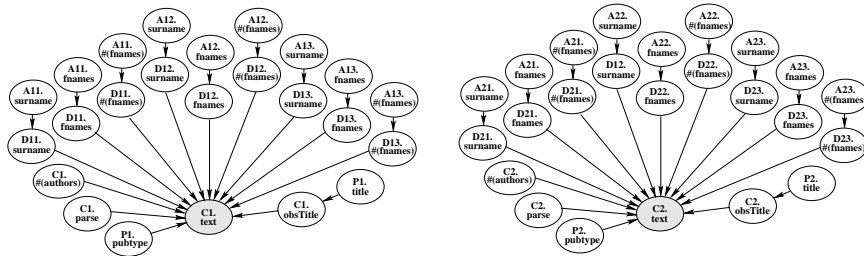

Figure 3: The Bayesian network equivalent to our RPM, assuming $C_1 \neq C_2$.

if they are distinct, there will be two sets. Thus, the possible worlds of our probability model may differ in the number of random variables, and there will be no single equivalent Bayesian network. The approach we have taken to this problem [10] is to extend the representation of a possible world so that it includes not only the basic attributes of a set of objects, but also the number of objects $n$ and an *identity clustering* $\iota$, that is, a mapping from terms in the language (such as $P_1$) to objects in the world. We are interested only in whether terms co-refer or not, so $\iota$ can be represented by a set of equivalence classes of terms. For example, if $P_1$ and $P_2$ are the only terms, and they co-refer, then $\iota$ is $\{\{P_1, P_2\}\}$; if they do not co-refer, then $\iota$ is $\{\{P_1\}, \{P_2\}\}$.

We define a probability model for the space of extended possible worlds by specifying the prior $P(n)$ and the conditional distribution $P(\iota|n)$. As in standard RPMs, we assume that the class of every instance is known. Hence, we can simplify these distributions further by factoring them by class, so that, e.g., $P(\iota) = \prod_{C \in \mathbf{C}} P(\iota_C)$. We then distinguish two cases:

- For some classes (such as the citations themselves), the unique names assumptions remains appropriate. Thus, we define $P(\iota_{Citation})$ to assign a probability of 1.0 to the one assignment where each citation object is unique.

- For classes such as *Paper* and *Author*, whose elements are subject to identity uncertainty, we specify $P(n)$ using a high-variance log-normal distribution.[3] Then we make appropriate uniformity assumptions to construct $P(\iota_C)$. Specifically, we assume that each paper is *a priori* equally likely to be cited, and that each author is *a priori* equally likely to write a paper. Here, "*a priori*" means prior to obtaining *any* information about the object in question, so the uniformity assumption is entirely reasonable. With these assumptions, the probability of an assignment $\iota_{C,k,m}$ that maps $k$ named instances to $m$ distinct objects, when $C$ contains $n$ objects, is given by

$$P(\iota_{C,k,m}) = \frac{n!}{(n-m)!} \frac{1}{n^k}$$

  When $n > m$, the world contains objects unreferenced by any of the terms. However, these filler objects are obviously irrelevant (if they affected the attributes of some named term, they would have been named as functions of that term.) Therefore, we never have to create them, or worry about their attribute values.

Our model assumes that the cardinalities and identity clusterings of the classes are independent of each other, as well as of the attribute values. We could remove these assumptions. For one, it would be straightforward to specify a class-wise dependency model for $n$ or $\iota$ using standard Bayesian network semantics, where the network nodes correspond to the cardinality attributes of the classes. E.g., it would be reasonable to let the total number of papers depend on the total number of authors. Similarly, we could allow $\iota$ to depend on the attribute values—e.g., the frequency of citations to a given paper might depend on the fame of the authors—provided we did not introduce cyclic dependencies.

## 4 The Probability Model

We will now fill in the details of the conditional probability models. Our priors over the "true" attributes are constructed off-line, using the following resources: the 1990 Census data on US names, a large A.I. BibTeX bibliography, and a hand-parsed collection of 500 citations. We learn several bigram models (actually, linear combinations of a bigram model and a unigram model): letter-based models of first names, surnames, and title words, as well as higher-level models of various parts of the citation string. More specifically, the values of *Author.fnames* and *Author.surname* are modeled as having a a 0.9 chance of being

drawn from the relevant US census file, and a $0.1$ chance of being generated using a bigram model learned from that file. The prior over *Paper.title*s is defined using a two-tier bigram model constructed using the bibliography, while the distributions over *Author.#(fnames)*, *Paper.#(authors)*, and *Paper.pubType* [4] are derived from our hand-parsed file. The conditional distributions of the "observed" variables given their true values (i.e., the corruption models of *Citation.obsTitle*, *AuthorAsCited.surname*, and *AuthorAsCited.fnames*) are modeled as noisy channels where each letter, or word, has a small probability of being deleted, or, alternatively, changed, and there is also a small probability of insertion. *AuthorAsCited.fnames* may also be abbreviated as an initial. The parameters of the corruption models are learnt online, using stochastic EM.

Let us now return to *Citation.parse*, which cannot be an observed variable, since citation parsing, or even citation subfield extraction, is an unsolved problem. It is therefore fortunate that our approach lets us handle uncertainty over parses so naturally. The state space of *Citation.parse* has two different components. First of all, it keeps track of the citation style, defined as the ordering of the author and title subfields, as well as the format in which the author names are written. The prior over styles is learned using our hand-segmented file. Secondly, it keeps track of the segmentation of *Citation.text*, which is divided into an author segment, a title segment, and three filler segments (one before, one after, and one in between.) We assume a uniform distribution over segmentations. *Citation.parse* greatly constrains *Citation.text*: the title segment of *Citation.text* must match the value of *Citation.obsTitle*, while its author segment must match the combined values of the simple attributes of *Citation.obsAuthors*. The distributions over the remaining three segments of *Citation.text* are defined using bigram models, with the model used for the final segment chosen depending on the publication type. These models were, once more, learned using our pre-segmented file.

## 5 INFERENCE

With the introduction of identity uncertainty, our model grows from a single Bayesian network to a collection of networks, one for each possible value of $\iota$. This collection can be rather large, since the number of ways in which a set can be partitioned grows very quickly with the size of the set. [5] Exact inference is, therefore, impractical. We use an approximate method based on Markov chain Monte Carlo.

### 5.1 MARKOV CHAIN MONTE CARLO

MCMC [13] is a well-known method for approximating an expectation over some distribution $\pi(x)$, commonly used when the state space of $x$ is too large to sum over. The weighted sum over the values of $x$ is replaced by a sum over samples from $\pi(x)$, which are generated using a Markov chain constructed to have $\pi(x)$ as a stationary distribution.

There are several ways of building up an appropriate Markov chain. In the *Metropolis–Hastings* method (M-H), transitions in the chain are constructed in two steps. First, a candidate next state $x'$ is generated from the current state $x$, using the (more or less arbitrary) *proposal distribution* $q(x'|x)$. The probability that the move to $x'$ is actually made is the *acceptance probability*, defined as $\alpha(x'|x) = min\left(1, \frac{\pi(x')q(x|x')}{\pi(x)q(x'|x)}\right)$.

Such a Markov chain will have the right stationary distribution $\pi(x)$ as long as $q$ is defined in such a way that the chain is ergodic. It is even possible to factor $q$ into separate proposals for various subsets of variables. In those situations, the variables that are not changed by the transition cancel in the ratio $\pi(x')/\pi(x)$, so the required calculation can be quite simple.

## 5.2 THE CITATION-MATCHING ALGORITHM

The state space of our MCMC algorithm is the space of all the possible worlds, where each possible world contains an identity clustering $\iota$, a set of class cardinalities $n$, and the values of all the basic attributes of all the objects. Since the $\iota$ is given in each world, the distribution over the attributes can be represented using a Bayesian network as described in Section 3. Therefore, the probability of a state is simply the product pf $P(n)$, $P(\iota)$, and the probability of the hidden attributes of the network.

Our algorithm uses a factored $q$ function. One of our proposals attempts to change $n$ using a simple random walk. The other suggests, first, a change to $\iota$, and then, values for all the hidden attributes of all the objects (or clusters in $\iota$) affected by that change. The algorithm for proposing a change in $\iota_C$ works as follows:

Select two clusters $a_1, a_2 \in \iota_C$[6]
Create two empty clusters $b_1$ and $b_2$
place each instance $i \in a_1 \cup a_2$ u.a.r. into $b_1$ or $b_2$
Propose $\iota'_C = \iota_C - \{a1, a2\} \cup \{b1, b2\}$

Given a proposed $\iota'_C$, suggesting values for the hidden attributes boils down to recovering their true values from (possibly) corrupt observations, e.g., guessing the true surname of the author currently known both as "Simth" and "Smith". Since our title and name noise models are symmetric, our basic strategy is to apply these noise models to one of the observed values. In the case of surnames, we have the additional resource of a dictionary of common names, so, some of the time, we instead pick one of the set of dictionary entries that are within a few corruptions of our observed names. (One must, of course, careful to account for this hybrid approach in our acceptance probability calculations.) Parses are handled differently: we preprocess each citation, organizing its plausible segmentations into a list ordered in terms of descending probability. At runtime, we simply sample from these discrete distributions. Since we assume that boundaries occur only at punctuation marks, and discard segmentations of probability $< 10^{-6}$, the lists are usually quite short. [7] The publication type variables, meanwhile, are not sampled at all. Since their range is so small, we sum them out.

## 5.3 SCALING UP

One of the acknowledged flaws of the MCMC algorithm is that it often fails to scale. In this application, as the number of papers increases, the simplest approach — one where the two clusters $a_1$ and $a_2$ are picked u.a.r — is likely to lead to many rejected proposals, as most pairs of clusters will have little in common. The resulting Markov chain will mix slowly. Clearly, we would prefer to focus our proposals on those pairs of clusters which are actually likely to exchange their instances. We have implemented an approach based on the efficient clustering algorithm of McCallum et al [9], where a cheap distance metric is used to preprocess a large dataset and fragment it into many *canopies*, or smaller, overlapping sets of elements that have a non-zero probability of matching. We do the same, using word-matching as our metric, and setting the thresholds to 0.5 and 0.2. Then, at runtime, our $q(x'|x)$ function proposes first a canopy $c$, and then a pair of clusters u.a.r. from $c$. ($q(x|x')$ is calculated by summing over all the canopies which contain any of the elements of the two clusters.)

## 6 EXPERIMENTAL RESULTS

We have applied the MCMC-based algorithm to the hand-matched datasets used in [1]. (Each of these datasets contains several hundred citations of machine learning papers, about half of them in clusters ranging in size from two to twenty-one citations.) We have also

| | Face 349 citations, 242 papers | Reinforcement 406 citations, 148 papers | Reasoning 514 citations, 296 papers | Constraint 295 citations, 199 papers |
|---|---|---|---|---|
| **Phrase matching** | *94%* | *79%* | *86%* | *89%* |
| **RPM + MCMC** | *97%* | *94%* | *96%* | *93%* |

Table 1: Results on four Citeseer data sets, for the text matching and MCMC algorithms. The metric used is the percentage of actual citation clusters recovered perfectly; for the MCMC-based algorithm, this is an average over all the MCMC-generated samples.

implemented their *phrase matching* algorithm, a greedy agglomerative clustering method based on a metric that measures the degrees to which the words and phrases of any two citations overlap. (They obtain their "phrases" by segmenting each citation at all punctuation marks, and then taking all the bigrams of all the segments longer than two words.) The results of our comparison are displayed in Figure 1, in terms of the Citeseer error metric. Clearly, the algorithm we have developed easily beats our implementation of phrase matching.

We have also applied our algorithm to a large set of citations referring to the textbook *Artificial Intelligence: A Modern Approach*. It clusters most of them correctly, but there are a couple of notable exceptions. Whenever several citations share the same set of unlikely errors, they are placed together in a separate cluster. This occurs because we do not currently model the fact that erroneous citations are often copied from reference list to reference list, which could be handled by extending the model to include a *copiedFrom* attribute. Another possible extension would be the addition of a *topic* attribute to both papers and authors: tracking the authors' research topics might enable the system to distinguish between similarly-named authors working in different fields. Generally speaking, we expect that relational probabilistic languages with identity uncertainty will be a useful tool for creating knowledge from raw data.

## Footnotes

[1]See `citeseer.nj.nec.com`. Citeseer is now known as ResearchIndex.

[2]Thus, uncertainty over whether the authors are ordered correctly can be modeled using probabilistic instance statements.

[3]Other models are possible; for example, in situations where objects appear and disappear, $P(\iota)$ can be modeled implicitly by specifying the arrival, transition, and departure rates [6].

[4]Publication types range over {article, conference paper, book, thesis, and tech report}

[5]This sequence is described by the Bell numbers, whose asymptotic behaviour is more than exponential.

[6]Note that if the same cluster is picked twice, it will probably be split.

[7]It would also be possible to sample directly from a model such as a hierarchical HMM

# References

[1] S. Lawrence, K. Bollacker, and C. Lee Giles. Autonomous citation matching. In *Agents*, 1999.

[2] I. Fellegi and A. Sunter. A theory for record linkage. In *JASA*, 1969.

[3] W. Cohen, H. Kautz, and D. McAllester. Hardening soft information sources. In *KDD*, 2000.

[4] Y. Bar-Shalom and T. E. Fortman. *Tracking and Data Association*. Academic Press, 1988.

[5] I. J. Cox and S. Hingorani. An efficient implementation and evaluation of Reid's multiple hypothesis tracking algorithm for visual tracking. In *IAPR-94*, 1994.

[6] H. Pasula, S. Russell, M. Ostland, and Y. Ritov. Tracking many objects with many sensors. In *IJCAI-99*, 1999.

[7] F. Dellaert, S. Seitz, C. Thorpe, and S. Thrun. Feature correspondence: A markov chain monte carlo approach. In *NIPS-00*, 2000.

[8] E. Charniak and R. P. Goldman. A Bayesian model of plan recognition. *AAAI*, 1993.

[9] A. McCallum, K. Nigam, and L. H. Ungar. Efficient clustering of high-dimensional data sets with application to reference matching. In *KDD-00*, 2000.

[10] H. Pasula and S. Russell. Approximate inference for first-order probabilistic languages. In *IJCAI-01*, 2001.

[11] A. Pfeffer. *Probabilistic Reasoning for Complex Systems*. PhD thesis, Stanford, 2000.

[12] A. Pfeffer and D. Koller. Semantics and inference for recursive probability models. In *AAAI/IAAI*, 2000.

[13] W.R. Gilks, S. Richardson, and D.J. Spiegelhalter. *Markov chain Monte Carlo in practice*. Chapman and Hall, London, 1996.
